# An LP View of the M-best MAP problem

**Menachem Fromer**     **Amir Globerson**
School of Computer Science and Engineering
The Hebrew University of Jerusalem
{fromer,gamir}@cs.huji.ac.il

## Abstract

We consider the problem of finding the $M$ assignments with maximum probability in a probabilistic graphical model. We show how this problem can be formulated as a linear program (LP) on a particular polytope. We prove that, for tree graphs (and junction trees in general), this polytope has a particularly simple form and differs from the marginal polytope in a single inequality constraint. We use this characterization to provide an approximation scheme for non-tree graphs, by using the set of spanning trees over such graphs. The method we present puts the $M$-best inference problem in the context of LP relaxations, which have recently received considerable attention and have proven useful in solving difficult inference problems. We show empirically that our method often finds the provably exact $M$ best configurations for problems of high tree-width.

A common task in probabilistic modeling is finding the assignment with maximum probability given a model. This is often referred to as the MAP (maximum a-posteriori) problem. Of particular interest is the case of MAP in graphical models, i.e., models where the probability factors into a product over small subsets of variables. For general models, this is an NP-hard problem [11], and thus approximation algorithms are required. Of those, the class of LP based relaxations has recently received considerable attention [3, 5, 18]. In fact, it has been shown that some problems (e.g., fixed backbone protein design) can be solved exactly via sequences of increasingly tighter LP relaxations [13].

In many applications, one is interested not only in the MAP assignment but also in the $M$ maximum probability assignments [19]. For example, in a protein design problem, we might be interested in the $M$ amino acid sequences that are most stable on a given backbone structure [2]. In cases where the MAP problem is tractable, one can devise tractable algorithms for the $M$ best problem [8, 19]. Specifically, for low tree-width graphs, this can be done via a variant of max-product [19]. However, when finding MAPs is not tractable, it is much less clear how to approximate the $M$ best case. One possible approach is to use loopy max-product to obtain approximate max-marginals and use those to approximate the $M$ best solutions [19]. However, this is largely a heuristic and does not provide any guarantees in terms of optimality certificates or bounds on the optimal values.

LP approximations to MAP do enjoy such guarantees. Specifically, they provide upper bounds on the MAP value and optimality certificates. Furthermore, they often work for graphs with large tree-width [13]. The goal of the current work is to leverage the power of LP relaxations to the $M$ best case. We begin by focusing on the problem of finding the second best solution. We show how it can be formulated as an LP over a polytope we call the "assignment-excluding marginal polytope". In the general case, this polytope may require an exponential number of inequalities, but we prove that when the graph is a tree it has a very compact representation. We proceed to use this result to obtain approximations to the second best problem, and show how these can be tightened in various ways. Next, we show how $M$ best assignments can be found by relying on algorithms for

second best assignments, and thus our results for the second best case can be used to devise an approximation algorithm for the $M$ best problem.

We conclude by applying our method to several models, showing that it often finds the *exact* $M$ best assignments.

# 1 The M-best MAP problem and its LP formulation

Consider a function on $n$ variables defined as:

$$f(x_1, \ldots, x_n; \boldsymbol{\theta}) = \sum_{ij \in E} \theta_{ij}(x_i, x_j) + \sum_{i \in V} \theta_i(x_i) \tag{1}$$

where $V$ and $E$ are the vertices and nodes of a graph $G$ with $n$ nodes. We shall be interested in the $M$ assignments with largest $f(\boldsymbol{x}; \boldsymbol{\theta})$ value.[1] Denote these by $\boldsymbol{x}^{(1)}, \ldots, \boldsymbol{x}^{(M)}$, so that $\boldsymbol{x}^{(1)}$ is the assignment that maximizes $f(\boldsymbol{x}; \boldsymbol{\theta})$, $\boldsymbol{x}^{(2)}$ is the $2^{nd}$ best assignment, etc.

The MAP problem (i.e., finding $\boldsymbol{x}^{(1)}$) can be formulated as an LP as follows [15]. Let $\boldsymbol{\mu}$ be a vector of distributions that includes $\{\mu_{ij}(x_i, x_j)\}_{ij \in E}$ over edge variables and $\{\mu_i(x_i)\}_{i \in V}$ over nodes. The set of $\boldsymbol{\mu}$ that arise from *some* joint distribution is known as the *marginal polytope* [15] and is denoted by $\mathcal{M}(G)$. Formally:

$$\mathcal{M}(G) = \{\boldsymbol{\mu} \mid \exists p(\boldsymbol{x}) \in \Delta \text{ s.t. } p(x_i, x_j) = \mu_{ij}(x_i, x_j) \;,\; p(x_i) = \mu_i(x_i)\}.$$

where $\Delta$ is the set of distributions on $\boldsymbol{x}$. The MAP problem can then be shown to be equivalent to the following LP:[2]

$$\max_{\boldsymbol{x}} f(\boldsymbol{x}; \boldsymbol{\theta}) = \max_{\boldsymbol{\mu} \in \mathcal{M}(G)} \boldsymbol{\mu} \cdot \boldsymbol{\theta} \;, \tag{2}$$

It can be shown that this LP always has a maximizing $\boldsymbol{\mu}$ that is a vertex of $\mathcal{M}(G)$ and is integral. Furthermore, this $\boldsymbol{\mu}$ corresponds to the MAP assignment $\boldsymbol{x}^{(1)}$. Although the number of variables in this LP is only $O(|E| + |V|)$, the difficulty comes from an exponential number of linear inequalities generally required to describe the marginal polytope $\mathcal{M}(G)$.

We shall find it useful to define a mapping between assignments $\boldsymbol{x}$ and integral vertices of the polytope. Given an integral vertex $\boldsymbol{v} \in \mathcal{M}(G)$, define $\boldsymbol{x}(\boldsymbol{v})$ to be the assignment that maximizes $v_i(x_i)$. And, given an assignment $\boldsymbol{z}$ define $\boldsymbol{v}(\boldsymbol{z})$ to be the integral vertex in $\mathcal{M}(G)$ corresponding to the assignment $\boldsymbol{z}$. Thus the LP in Eq. 2 will be maximized by $\boldsymbol{v}(\boldsymbol{x}^{(1)})$.

One simple outer bound of the marginal polytope is the local polytope $\mathcal{M}_L(G)$, which only enforces pairwise constraints between variables:

$$\mathcal{M}_L(G) = \left\{ \boldsymbol{\mu} \geq 0 \left| \sum_{x_j} \mu_{ij}(x_i, x_j) = \mu_i(x_i), \sum_{x_i} \mu_{ij}(x_i, x_j) = \mu_j(x_j), \sum_{x_i} \mu_i(x_i) = 1 \right. \right\} \tag{3}$$

The LP relaxation is then to maximize $\boldsymbol{\mu} \cdot \boldsymbol{\theta}$ where $\boldsymbol{\mu} \in \mathcal{M}_L(G)$. For tree structured graphs, $\mathcal{M}_L(G) = \mathcal{M}(G)$ [15] and thus the LP relaxation yields the exact MAP $\boldsymbol{x}^{(1)}$.

# 2 An LP Formulation for the $2^{nd}$-best MAP

Assume we found the MAP assignment $\boldsymbol{x}^{(1)}$ and are now interested in finding $\boldsymbol{x}^{(2)}$. Is there a simple LP whose solution yields $\boldsymbol{x}^{(2)}$? We begin by focusing on the case where $G$ is a tree so that the local LP relaxation is exact. We first treat the case of a connected tree.

To construct an LP whose solution is $\boldsymbol{x}^{(2)}$, a natural approach is to use the LP for $\boldsymbol{x}^{(1)}$ (i.e., the LP in Eq. 2) but somehow eliminate the solution $\boldsymbol{x}^{(1)}$ using additional constraints. This, however, is somewhat trickier than it sounds. The key difficulty is that the new constraints should not generate fractional vertices, so that the resulting LP is still exact.

We begin by defining the polytope over which we need to optimize in order to obtain $\boldsymbol{x}^{(2)}$.

**Definition 1.** *The assignment-excluding marginal polytope is defined as:*

$$\hat{\mathcal{M}}(G, \boldsymbol{z}) = \{\boldsymbol{\mu} \mid \exists p(\boldsymbol{x}) \in \Delta \quad s.t. \quad p(\boldsymbol{z}) = 0, p(x_i, x_j) = \mu_{ij}(x_i, x_j), p(x_i) = \mu_i(x_i)\}. \quad (4)$$

$\hat{\mathcal{M}}(G, \boldsymbol{z})$ *is simply the convex hull of all (integral) vectors $\boldsymbol{v}(\boldsymbol{x})$ for $\boldsymbol{x} \neq \boldsymbol{z}$.*

The following result shows that optimizing over $\hat{\mathcal{M}}(G, \boldsymbol{x}^{(1)})$ will yield the second best solution $\boldsymbol{x}^{(2)}$, so that we refer to $\hat{\mathcal{M}}(G, \boldsymbol{x}^{(1)})$ as the *second-best marginal polytope*.

**Lemma 1.** *The $2^{nd}$ best solution is obtained via the following LP:*
$\max_{\boldsymbol{x} \neq \boldsymbol{x}^{(1)}} f(\boldsymbol{x}; \boldsymbol{\theta}) = \max_{\boldsymbol{\mu} \in \hat{\mathcal{M}}(G, \boldsymbol{x}^{(1)})} \boldsymbol{\mu} \cdot \boldsymbol{\theta}$. *Furthermore, the $\boldsymbol{\mu}$ that maximizes the LP on the right is integral and corresponds to the second-best MAP assignment $\boldsymbol{x}^{(2)}$.*

The proof is similar to that of Eq. 2: instead of optimizing over $\boldsymbol{x}$, we optimize over distributions $p(\boldsymbol{x})$, while enforcing that $p(\boldsymbol{x}^{(1)}) = 0$ so that $\boldsymbol{x}^{(1)}$ is excluded from the maximization.

The key question which we now address is how to obtain a simple characterization of $\hat{\mathcal{M}}(G, \boldsymbol{z})$. Intuitively, it would seems that $\hat{\mathcal{M}}(G, \boldsymbol{z})$ should be "similar" to $\mathcal{M}(G)$, such that it can be described as $\mathcal{M}(G)$ plus some constraints that "block" the assignment $\boldsymbol{z}$. To illustrate the difficulty in finding such "blocking" constraints, consider the following constraint, originally suggested by Santos [10]: $\sum_i \mu_i(z_i) \leq n - 1$. This inequality is not satisfied by $\boldsymbol{\mu} = \boldsymbol{v}(\boldsymbol{z})$ since $\boldsymbol{v}(\boldsymbol{z})$ attains the value $n$ for the LHS of the above. Furthermore, for any $\boldsymbol{x} \neq \boldsymbol{z}$ and $\boldsymbol{\mu} = \boldsymbol{v}(\boldsymbol{x})$, the LHS would be $n-1$ or less. Thus, this inequality separates $\boldsymbol{v}(\boldsymbol{z})$ from all other integral vertices. One might conclude that we can define $\hat{\mathcal{M}}(G, \boldsymbol{z})$ by adding this inequality to $\mathcal{M}(G)$. The difficulty is that the resulting polytope has fractional vertices,[3] and maximizing over it won't generally yield an integral solution.

It turns out that there is a different inequality that does yield an exact characterization of $\hat{\mathcal{M}}(G, \boldsymbol{z})$ when $G$ is a tree. We now define this inequality and state our main theorem.

**Definition 2.** *Consider the functional $I(\boldsymbol{\mu}, \boldsymbol{z})$ (which is linear in $\boldsymbol{\mu}$):*

$$I(\boldsymbol{\mu}, \boldsymbol{z}) = \sum_i (1 - d_i) \mu_i(z_i) + \sum_{ij \in E} \mu_{ij}(z_i, z_j) \quad (5)$$

*where $d_i$ is the degree of node $i$ in the tree graph $G$.*

**Theorem 1.** *Adding the single inequality $I(\boldsymbol{\mu}, \boldsymbol{z}) \leq 0$ to $\mathcal{M}(G)$ yields $\hat{\mathcal{M}}(G, \boldsymbol{z})$.*

$$\hat{\mathcal{M}}(G, \boldsymbol{z}) = \{\boldsymbol{\mu} \mid \boldsymbol{\mu} \in \mathcal{M}(G), \quad I(\boldsymbol{\mu}, \boldsymbol{z}) \leq 0 \} \quad (6)$$

The theorem is proved in the appendix. Taken together with Lemma 1, it implies that $\boldsymbol{x}^{(2)}$ may be obtained via an LP that is very similar to the MAP-LP, but has an additional constraint. We note the interesting similarity between $I(\boldsymbol{\mu}, \boldsymbol{z})$ and the Bethe entropy [20]. The only difference is that in Bethe, $\mu_i, \mu_{ij}$ are replaced by $H(X_i), H(X_i, X_j)$ respectively.[4]

The theorem also generalizes to the case where $G$ is not a tree, but we have a junction tree for $G$. In this case, the theorem still holds if we define a generalized $I(\boldsymbol{\mu}, \boldsymbol{z})$ inequality as:

$$\sum_{S \in \mathcal{S}} (1 - d_S) \mu_S(z_S) + \sum_{C \in \mathcal{C}} \mu_C(z_C) \leq 0 \quad (7)$$

where $\mathcal{C}$ and $\mathcal{S}$ are the junction tree cliques and their separators, respectively, and $d_S$ is the number of cliques that intersect on separator $S$. In this case, the marginal polytope should enforce consistency between marginals $\mu_C(z_C)$ and their separators $\mu_S(z_S)$. However, such a characterization requires variables whose cardinality is exponential in the tree-width and is thus tractable only for graphs of low tree-width. In the next section, we address approximations for general graphs.

A corresponding result exists for the case when $G$ is a forest. In this case, the inequality in Eq. 6 is modified to: $I(\boldsymbol{\mu}, \boldsymbol{z}) \leq |P| - 1$, where $|P|$ denotes the number of connected components of $G$. Interestingly, for a graph without edges, this gives the Santos inequality.

# 3    $2^{nd}$ best LPs for general graphs - Spanning tree inequalities

When the graph $G$ is not a tree, the marginal polytope $\mathcal{M}(G)$ generally requires an exponential number of inequalities. However, as mentioned above, it does have an exact description in terms of marginals over cliques and separators of a junction tree. Given such marginals on junction tree cliques, we also have an exact characterization of $\hat{\mathcal{M}}(G, \boldsymbol{z})$ via the constraint in Eq. 7. However, in general, we cannot afford to be exponential in tree-width. Thus a common strategy [15] is to replace $\mathcal{M}(G)$ with an outer bound that enforces consistency between marginals on overlapping sets of variables. The simplest example is $\mathcal{M}_L(G)$ in Eq. 3. In what follows, we describe an outer-bound approximation scheme for $\hat{\mathcal{M}}(G, \boldsymbol{z})$. We use $\mathcal{M}_L(G)$ as the approximation for $\mathcal{M}(G)$ (more generally $\mathcal{M}_L(G)$ can enforce consistency between any set of small regions, e.g., triplets). When $G$ is not a tree, the linear constraint in Eq. 6 will no longer suffice to derive $\hat{\mathcal{M}}(G, \boldsymbol{z})$. Moreover, direct application of the inequality will incorrectly remove some integral vertices. An alternative approach is to add inequalities that separate $\boldsymbol{v}(\boldsymbol{z})$ from the other integral vertices. This will serve to eliminate more and more fractional vertices, and if enough constraints are added, this may result in an integral solution. One obvious family of such constraints are those corresponding to spanning trees in $G$ and have the form of Eq. 5.

**Definition 3.** *Consider any $T$ that is a spanning tree of $G$. Define the functional $I^T(\boldsymbol{\mu}, \boldsymbol{z})$:*

$$I^T(\boldsymbol{\mu}, \boldsymbol{z}) = \sum_i (1 - d_i^T)\mu_i(z_i) + \sum_{ij \in T} \mu_{ij}(z_i, z_j) \qquad (8)$$

*where $d_i^T$ is the degree of $i$ in $T$. We refer to $I^T(\boldsymbol{\mu}, \boldsymbol{z}) \leq 0$ as a* **spanning tree inequality**.

For any sub-tree $T$ of $G$, the corresponding spanning tree inequality separates the vertex $\boldsymbol{v}(\boldsymbol{z})$ from the other vertices. This can be shown via similar arguments as in the proof of Theorem 1. Note, however, that the resulting polytope may still have fractional vertices.

The above argument shows that any spanning tree provides a separating inequality for $\hat{\mathcal{M}}(G, \boldsymbol{z})$. In principle, we would like to use as many such inequalities as possible.

**Definition 4.** *The spanning tree assignment-excluding marginal polytope is defined as:*

$$\hat{\mathcal{M}}_L^{\mathcal{ST}}(G, \boldsymbol{z}) = \left\{ \boldsymbol{\mu} \mid \boldsymbol{\mu} \in \mathcal{M}_L(G), \quad \forall \ tree \ T \subseteq E \quad I^T(\boldsymbol{\mu}, \boldsymbol{z}) \leq 0 \right\} \qquad (9)$$

*where the $\mathcal{ST}$ notation indicates the inclusion of all spanning tree inequalities for $G$.*[5]

Thus, we would actually like to perform the following optimization problem: $\displaystyle\max_{\boldsymbol{\mu} \in \hat{\mathcal{M}}_L^{\mathcal{ST}}(G, \boldsymbol{z})} \boldsymbol{\mu} \cdot \boldsymbol{\theta}$

as an approximation to optimization over $\hat{\mathcal{M}}(G, \boldsymbol{z})$; i.e., we seek the optimal $\boldsymbol{\mu}$ subject to all spanning tree inequalities for $G$ with the ambition that this $\boldsymbol{\mu}$ be integral and thus provide the non-$\boldsymbol{z}$ MAP assignment, with a certificate of optimality.

Although the number of spanning trees is exponential in $n$, it turns out that *all* spanning inequalities can be used in practice. One way to achieve this is via a cutting plane algorithm [12] that finds the most violated spanning tree inequality and adds it to the LP. To implement this efficiently, we note that for a particular $\boldsymbol{\mu}$ and a spanning tree $T$, the value of $I^T(\boldsymbol{\mu}, \boldsymbol{z})$ can be decomposed into a sum over the edges in $T$ (and a $T$-independent constant):

$$I^T(\boldsymbol{\mu}, \boldsymbol{z}) = \sum_{ij \in T} \left[ \mu_{ij}(z_i, z_j) - \mu_i(z_i) - \mu_j(z_j) \right] + \sum_i \mu_i(z_i) \qquad (10)$$

The tree maximizing the above is the maximum-weight spanning tree with edge-weights $w_{ij} = \mu_{ij}(z_i, z_j) - \mu_i(z_i) - \mu_j(z_j)$. It can thus be found efficiently.

The cutting plane algorithm proceeds as follows. We start by adding an arbitrary spanning tree. Then, as long as the optimal $\boldsymbol{\mu}$ is fractional, we find the spanning tree inequality that $\boldsymbol{\mu}$ most violates (where this is implemented via the maximum-weight spanning tree). This constraint will necessarily remove $\boldsymbol{\mu}$ from the polytope. If there are no violated inequalities

but $\boldsymbol{\mu}$ is still fractional, then spanning tree inequalities do not suffice to find an integral solution (but see below on hypertree constraints to add in this case). In practice, we found that only a relatively small number of inequalities are needed to successfully yield an integral solution, or determine that all such inequalities are already satisfied.

An alternative approach for solving the all spanning-tree problem is to work via the dual. The dual variables roughly correspond to points in the spanning tree polytope [16], optimization over which can be done in polynomial time, e.g., via the ellipsoid algorithm. We do not pursue this here since the cutting plane algorithm performed well in our experiments.

As mentioned earlier, we can exactly characterize $\hat{\mathcal{M}}(G, \boldsymbol{z})$ using Eq. 7, albeit at a cost exponential in the tree-width of the graph. A practical compromise would be to use inequalities over clique trees of $G$, where the cliques are relatively small, e.g., triplets. The corresponding constraint (Eq. 7 with the small cliques and their separators) will necessarily separate $\boldsymbol{v}(\boldsymbol{z})$ from the other integral vertices. Finding the maximally violated such inequality is an NP-hard problem, equivalent to a prize collecting Steiner tree problem, but recent work has found that such problems are often exactly solvable in practice [7]. It thus might be practical to include all such trees as constraints using a cutting plane algorithm.

# 4 From $2^{nd}$-best to M-best

Thus far, we only dealt with the $2^{nd}$ best case. As we show now, it turns out that the $2^{nd}$-best formalism can be used to devise an algorithm for $M$ best. We begin by describing an algorithm for the exact $M$ best and then show how it can be used to approximate those via the approximations for $2^{nd}$ best described above. Fig. 1 describes our scheme, which we call **P**artitioning for **E**numerating **S**olutions (or PES) for solving the $M$ best problem. The scheme is general and only assumes that MAP-"like" problems can be solved. It is inspired by several pre-existing $M$ best solution schemes [4, 6, 8, 19] but differs from them in highlighting the role of finding a second best solution within a given subspace.

---

**for** $m \leftarrow 1$ **to** $M$ **do**

    **if** $m = 1$ **then**

        Run MAP solver to obtain the best assignment: $\boldsymbol{x}^{(1)} \equiv \arg\max f(\boldsymbol{x}; \boldsymbol{\theta})$

        CONSTRAINTS$^1 \leftarrow \emptyset$

    **else**

        $k \longleftarrow \underset{k' \in \{1, \ldots, m-1\}}{\arg\max} f(\boldsymbol{y}^{(k')}; \boldsymbol{\theta})$ // sub-space containing $m^{th}$ best assignment

        $\boldsymbol{x}^{(m)} \leftarrow \boldsymbol{y}^{(k)}$ // $m^{th}$ best assignment

        // A variable choice that distinguishes $\boldsymbol{x}^{(m)}$ from $\boldsymbol{x}^{(k)}$:

        $(v, a) \leftarrow$ any member of the set $\{(i, x_i^{(m)}) : x_i^{(m)} \neq x_i^{(k)}\}$

        CONSTRAINTS$^m \leftarrow$ CONSTRAINTS$^k \cup \{x_v = a\}$ // Eliminate $\boldsymbol{x}^{(k)}$ (as MAP) from subspace $m$

        CONSTRAINTS$^k \leftarrow$ CONSTRAINTS$^k \cup \{x_v \neq a\}$ // Eliminate $\boldsymbol{x}^{(m)}$ (as $2^{nd}$-best) from subspace $k$

        $\boldsymbol{y}^{(k)} \leftarrow$ CalcNextBestSolution(CONSTRAINTS$^k$, $\boldsymbol{x}^{(k)}$)

    **end**

    $\boldsymbol{y}^{(m)} \leftarrow$ CalcNextBestSolution(CONSTRAINTS$^m$, $\boldsymbol{x}^{(m)}$)

**end**

**return** $\{\boldsymbol{x}^{(m)}\}_{m=1}^M$

---

/* Find next best solution in sub-space defined by CONSTRAINTS */

**Function** CalcNextBestSolution(CONSTRAINTS, $\boldsymbol{x}^{(*)}$)

    // $\boldsymbol{x}^{(*)}$ is the MAP in the sub-space defined by CONSTRAINTS:

    Run MAP solver to obtain the second-best solution: $\boldsymbol{y} \equiv \underset{\boldsymbol{x} \neq \boldsymbol{x}^{(*)}, \text{CONSTRAINTS}}{\arg\max} f(\boldsymbol{x}; \boldsymbol{\theta})$, and return $\boldsymbol{y}$.

**end**

Figure 1: Pseudocode for the PES algorithm.

The modus operandi of the PES algorithm is to efficiently partition the search space while systematically excluding all previously determined assignments. Significantly, any MAP

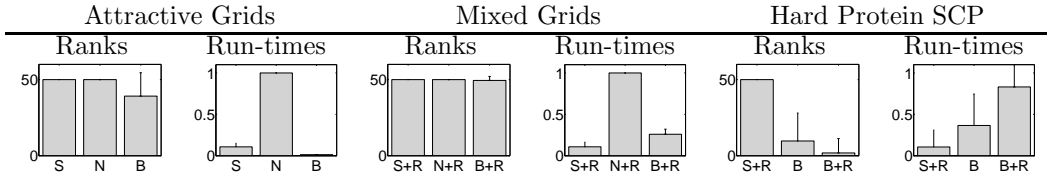

Figure 2: Number of best ranks and normalized run-times for the attractive and mixed grids, and the more difficult protein SCP problems. S, N, and B denote the STRIPES, Nilsson, and BMMF algorithms. Algorithms marked with +R denote that regions of variables were added for those runs.

solver can be plugged into it, on the condition that it is capable of solving the $\arg\max$ in the `CalcNextBestSolution` subroutine. The correctness of PES can be shown by observing that at the $M^{th}$ stage, all previous best solutions are excluded from the optimization and no other assignment is excluded. Of note, this simple partitioning scheme is possible due to the observation that the first-best and second-best MAP assignments must differ in the assignment of at least one variable in the graph.

The main computational step of the PES algorithm is to maximize $f(\boldsymbol{x};\boldsymbol{\theta})$ subject to $\boldsymbol{x} \neq \boldsymbol{x}^{(*)}$ and $\boldsymbol{x} \in \text{CONSTRAINTS}$ (see the `CalcNextBestSolution` subroutine). The CONSTRAINTS set merely enforces that some of the coordinates of $\boldsymbol{x}$ are either equal to or different from specified values.[6] Within the LP, these can be enforced by setting $\mu_i(x_i = a) = 1$ or $\mu_i(x_i = a) = 0$. It can be shown that if one optimizes $\boldsymbol{\mu} \cdot \boldsymbol{\theta}$ with these constraints and $\boldsymbol{\mu} \in \hat{\mathcal{M}}(G, \boldsymbol{x}^{(*)})$, the solution is integral. Thus, the only element requiring approximation in the general case is the description of $\hat{\mathcal{M}}(G, \boldsymbol{x}^{(*)})$. We choose as this approximation the polytope $\hat{\mathcal{M}}_L^{\mathcal{ST}}(G, \boldsymbol{x}^{(*)})$ in Eq. 9. We call the resulting approximation algorithm **S**panning **TR**ee **I**nequalities and **P**artitioning for **E**numerating **S**olutions, or STRIPES. In the next section, we evaluate this scheme experimentally.

## 5 Experiments

We compared the performance of STRIPES to the BMMF algorithm [19] and the Lawler/Nilsson algorithm [6, 8]. Nilsson's algorithm is equivalent to PES where the $2^{nd}$ best assignment is obtained from maximizations within $O(n)$ partitions, so that its run-time is $O(n)$ times the cost of finding a single MAP. Here we approximated each MAP with its LP relaxation (as in STRIPES), so that both STRIPES and Nilsson come with certificates of optimality when their LP solutions are integral. BMMF relies on loopy BP to approximate the $M$ best solutions.[7] We used $M = 50$ in all experiments. To compare the algorithms, we pooled all their solutions, noting the 50 top probabilities, and then counted the fraction of these that any particular algorithm found (its solution rank). For run-time comparisons, we normalized the times by the longest-running algorithm for each example.

We begin by considering pairwise MRFs on binary grid graphs of size $10 \times 10$. In the first experiment, we used an Ising model with attractive (submodular) potentials, a setting in which the pairwise LP relaxation is exact [14]. For each grid edge $ij$, we randomly chose $J_{ij} \in [0, 0.5]$, and local potentials were randomized in the range $\pm 0.5$. The results for 25 graphs are shown in Fig. 2. Both the STRIPES and Nilsson algorithms obtained the 50 optimal solutions (as learned from their optimality certificates), while BMMF clearly fared less well for some of the graphs. While the STRIPES algorithm took < 0.5 to 2 minutes to run, the Nilsson algorithm took around 13 minutes. On the other hand, BMMF was quicker, taking around 10 seconds per run, while failing to find a significant portion of the top solutions. Overall, the STRIPES algorithm was required to employ up to 19 spanning tree inequalities per calculation of second-best solution.

Next, we studied Ising models with mixed interaction potentials (with $J_{ij}$ and the local potentials randomly chosen in $[-0.5, 0.5]$). For almost all of the 25 models, all three algorithms were not able to successfully find the top solutions. Thus, we added regions of triplets (two for every grid face) to tighten the LP relaxation (for STRIPES and Nilsson) and to perform GBP instead of BP (for BMMF). This resulted in STRIPES and Nilsson always provably finding the optimal solutions, and BMMF mostly finding these solutions (Fig. 2). For these more difficult grids, however, STRIPES was the fastest of the algorithms, taking 0.5 - 5 minutes. On the other hand, the Nilsson and BMMF algorithms took 18 minutes and 2.5 - 7 minutes, respectively. STRIPES added up to 23 spanning tree inequalities per iteration.

The protein side-chain prediction (SCP) problem is to to predict the placement of amino acid side-chains given a protein backbone [2, 18]. Minimization of a protein energy function corresponds to finding a MAP assignment for a pairwise MRF [19]. We employed the dataset of [18] (up to 45 states per variable, mean approximate tree-width 50), running all algorithms to calculate the optimal side-chain configurations. For 315 of 370 problems in the dataset, the first MAP solution was obtained directly as a result of the LP relaxation having an integral solution ("easy" problems). STRIPES provably found the subsequent top 50 solutions within 4.5 hours for all but one of these cases (up to 8 spanning trees per calculation), and BMMF found the same 50 solutions for each case within 0.5 hours; note that only STRIPES provides a certificate of optimality for these solutions. On the other hand, only for 146 of the 315 problems was the Nilsson method able to complete within five days; thus, we do not compare its performance here. For the remaining 55 ("hard") problems (Fig. 2), we added problem-specific triplet regions using the MPLP algorithm [13]. We then ran the STRIPES algorithm to find the optimal solutions. Surprisingly, it was able to exactly find the 50 top solutions for all cases, using up to 4 standard spanning tree inequalities per second-best calculation. The STRIPES run-times for these problems ranged from 6 minutes to 23 hours. On the other hand, whether running BMMF without these regions (BP) or with the regions (GBP), it did not perform as well as STRIPES in terms of the number of high-ranking solutions or its speed. To summarize, STRIPES provably found the top 50 solutions for 369 of the 370 protein SCP problems.

# 6 Conclusion

In this work, we present a novel combinatorial object $\hat{\mathcal{M}}(G, \boldsymbol{z})$ and show its utility in obtaining the $M$ best MAP assignments. We provide a simple characterization of it for tree structured graphs, and show how it can be used for approximations in non-tree graphs. As with the marginal polytope, many interesting questions arise about the properties of $\hat{\mathcal{M}}(G, \boldsymbol{z})$. For example, in which non-tree cases can we provide a compact characterization (e.g., as for the cut-polytope for planar graphs [1]). Another compelling question is in which problems the spanning tree inequalities are provably optimal.

An interesting generalization of our method is to predict diverse solutions satisfying some local measure of "distance" from each other, e.g., as in [2].

Here we studied the polytope that results from excluding one assignment. An intriguing question is to characterize the polytope that excludes $M$ assignments. We have found that it does not simply correspond to adding $M$ constraints $I(\boldsymbol{\mu}, \boldsymbol{z}^i) \leq 0$ for $i = 1, \ldots, M$, so its geometry is apparently more complicated than that of $\hat{\mathcal{M}}(G, \boldsymbol{z})$.

Here we used LP solvers to solve for $\boldsymbol{\mu}$. Such generic solvers could be slow for large-scale problems. However, in recent years, specialized algorithms have been suggested for solving MAP-LP relaxations [3, 5, 9, 17]. These use the special form of the constraints to obtain local-updates and more scalable algorithms. We intend to apply these schemes to our method. Finally, our empirical results show that our method indeed leverages the power of LP relaxations and yields exact $M$ best optimal solutions for problems with large tree-width.

### Acknowledgements

We thank Nati Linial for his helpful discussions and Chen Yanover and Talya Meltzer for their insight and help in running BMMF. We also thank the anonymous reviewers for their useful advice.

## A   Proof of Theorem 1

Recall that for any $\boldsymbol{\mu} \in \mathcal{M}(G)$, there exists a probability density $p(\boldsymbol{x})$ s.t. $\boldsymbol{\mu} = \sum_{\boldsymbol{x}} p(\boldsymbol{x}) \boldsymbol{v}(\boldsymbol{x})$. Denote $p_{\boldsymbol{\mu}}(\boldsymbol{z})$ as the minimal value of $p(\boldsymbol{z})$ among all $p(\boldsymbol{x})$ that give $\boldsymbol{\mu}$. We prove that $p_{\boldsymbol{\mu}}(\boldsymbol{z}) = \max(0, I(\boldsymbol{\mu}, \boldsymbol{z}))$, from which the theorem follows (since $p_{\boldsymbol{\mu}}(\boldsymbol{z}) = 0$ iff $\boldsymbol{\mu} \in \hat{\mathcal{M}}(G, \boldsymbol{z})$).

The proof is by induction on $n$. For $n = 1$, the node has degree 0, so $I(\boldsymbol{\mu}, \boldsymbol{z}) = \mu_1(z_1)$. Clearly, $p_{\boldsymbol{\mu}}(\boldsymbol{z}) = \mu_1(z_1)$, so $p_{\boldsymbol{\mu}}(\boldsymbol{z}) = I(\boldsymbol{\mu}, \boldsymbol{z})$. For $n > 1$, there must exist a leaf in $G$ (assume that its index is $n$ and its neighbor's is $n-1$). Denote $\hat{G}$ as the tree obtained by removing node $n$ and its edge with $n-1$. For any assignment $\boldsymbol{x}$, denote $\hat{\boldsymbol{x}}$ as the corresponding sub-assignment for the first $n-1$ variables. Also, any $\boldsymbol{\mu}$ can be derived by adding appropriate coordinates to a unique $\hat{\boldsymbol{\mu}} \in \mathcal{M}(\hat{G})$. For an integral vertex $\boldsymbol{\mu} = \boldsymbol{v}(\boldsymbol{x})$, denote its projected $\hat{\boldsymbol{\mu}}$ as $\hat{\boldsymbol{v}}(\hat{\boldsymbol{x}})$. Denote by $\hat{I}(\hat{\boldsymbol{\mu}}, \hat{\boldsymbol{z}})$ the functional in Eq. 5 applied to $\hat{G}$. For any $\boldsymbol{\mu}$ and its projected $\hat{\boldsymbol{\mu}}$, it can be seen that:

$$I(\boldsymbol{\mu}, \boldsymbol{z}) = \hat{I}(\hat{\boldsymbol{\mu}}, \hat{\boldsymbol{z}}) - \alpha \tag{11}$$

where we define $\alpha = \sum_{x_n \neq z_n} \mu_{n-1,n}(z_{n-1}, x_n)$ (so $0 \leq \alpha \leq 1$). The inductive assumption gives a $\hat{p}(\hat{\boldsymbol{x}})$ that has marginals $\hat{\boldsymbol{\mu}}$ *and* also $\hat{p}(\hat{\boldsymbol{z}}) = \max(0, \hat{I}(\hat{\boldsymbol{\mu}}, \hat{\boldsymbol{z}}))$. We next use $\hat{p}(\hat{\boldsymbol{x}})$ to construct a $p(\boldsymbol{x})$ that has marginals $\boldsymbol{\mu}$ and the desired minimal $p_{\boldsymbol{\mu}}(\boldsymbol{z})$. Consider three cases:

**I.** $I(\boldsymbol{\mu}, \boldsymbol{z}) \leq 0$ and $\hat{I}(\hat{\boldsymbol{\mu}}, \hat{\boldsymbol{z}}) \leq 0$. From the inductive assumption, $\hat{p}_{\hat{\boldsymbol{\mu}}}(\hat{\boldsymbol{z}}) = 0$, so we define:

$$p(\boldsymbol{x}) = \hat{p}(\hat{\boldsymbol{x}}) \frac{\mu_{n-1,n}(x_{n-1}, x_n)}{\mu_{n-1}(x_{n-1})} \tag{12}$$

which indeed marginalizes to $\boldsymbol{\mu}$, and $p(\boldsymbol{z}) = 0$ so that $p_{\boldsymbol{\mu}}(\boldsymbol{z}) = 0$ as required. If $\mu_{n-1}(x_{n-1}) = 0$, then $\hat{p}(\hat{\boldsymbol{x}})$ is necessarily 0, in which case we define $p(\boldsymbol{x}) = 0$. Note that this construction is identical to that used in proving that $\mathcal{M}_L(G) = \mathcal{M}(G)$ for a tree graph $G$.

**II.** $I(\boldsymbol{\mu}, \boldsymbol{z}) > 0$. Based on Eq. 11 and $\alpha \geq 0$, we have $\hat{I}(\hat{\boldsymbol{\mu}}, \hat{\boldsymbol{z}}) > 0$. Applying the inductive assumption to $\hat{\boldsymbol{\mu}}$, we obtain $\hat{I}(\hat{\boldsymbol{\mu}}, \hat{\boldsymbol{z}}) = \hat{p}_{\hat{\boldsymbol{\mu}}}(\hat{\boldsymbol{z}}) > 0$. Now, define $p(\boldsymbol{x})$ so that $p(\boldsymbol{z}) = I(\boldsymbol{\mu}, \boldsymbol{z})$:

| $x_l,\ l \leq n-2$ | $\delta(x_{n-1} = z_{n-1})$ | $\delta(x_n = z_n)$ | $p(\boldsymbol{x})$ |
|---|---|---|---|
| no constraint | 0 | no constraint | As in Eq. 12 |
| $\exists\, l\ \ x_l \neq z_l$ | 1 | 0 | 0 |
| | | 1 | $\hat{p}(\hat{\boldsymbol{x}})$ |
| $\forall\, l\ \ x_l = z_l$ | 1 | 0 | $\mu_{n-1,n}(z_{n-1}, x_n)$ |
| | | 1 | $I(\boldsymbol{\mu}, \boldsymbol{z})$ |

Simple algebra shows that $p(\boldsymbol{x})$ is non-negative and has $\boldsymbol{\mu}$ as marginals. We now show that $p(\boldsymbol{z})$ is minimal. Based on the inductive assumption and Eq. 11, it can easily be shown that $I(\boldsymbol{v}(\boldsymbol{z}), \boldsymbol{z}) = 1$, $I(\boldsymbol{v}(\boldsymbol{x}), \boldsymbol{z}) \leq 0$ for $\boldsymbol{x} \neq \boldsymbol{z}$. For any $p(\boldsymbol{x})$ s.t. $\boldsymbol{\mu} = \sum_{\boldsymbol{x}} p(\boldsymbol{x}) \boldsymbol{v}(\boldsymbol{x})$, from linearity, $I(\boldsymbol{\mu}, \boldsymbol{z}) = p(\boldsymbol{z}) + \sum_{\boldsymbol{x} \neq \boldsymbol{z}} p(\boldsymbol{x}) I(\boldsymbol{v}(\boldsymbol{x}), \boldsymbol{z}) \leq p(\boldsymbol{z})$ (since $I(\boldsymbol{v}(\boldsymbol{x}), \boldsymbol{z}) \leq 0$ for $\boldsymbol{x} \neq \boldsymbol{z}$). Since the $p(\boldsymbol{z})$ we define achieves this lower bound, it is clearly minimal.

**III.** $I(\boldsymbol{\mu}, \boldsymbol{z}) \leq 0$ but $\hat{I}(\hat{\boldsymbol{\mu}}, \hat{\boldsymbol{z}}) > 0$. Applying the inductive assumption to $\hat{\boldsymbol{\mu}}$, we see that $\hat{p}_{\hat{\boldsymbol{\mu}}}(\hat{\boldsymbol{z}}) = \hat{I}(\hat{\boldsymbol{\mu}}, \hat{\boldsymbol{z}}) > 0$; Eq. 11 implies $\alpha - \hat{I}(\hat{\boldsymbol{\mu}}, \hat{\boldsymbol{z}}) \geq 0$. Define $\beta = \mu_{n-1}(z_{n-1}) - \hat{p}_{\hat{\boldsymbol{\mu}}}(\hat{\boldsymbol{z}})$, which is non-negative since $\mu_{n-1}(z_{n-1}) = \hat{\mu}_{n-1}(\hat{z}_{n-1})$ and $\hat{p}$ marginalizes to $\hat{\boldsymbol{\mu}}$. Define $p(\boldsymbol{x})$ as:

| $x_l,\ l \leq n-2$ | $\delta(x_{n-1} = z_{n-1})$ | $\delta(x_n = z_n)$ | $p(\boldsymbol{x})$ |
|---|---|---|---|
| no constraint | 0 | no constraint | As in Eq. 12 |
| $\exists\, l\ \ x_l \neq z_l$ | 1 | 0 | $\hat{p}(\hat{\boldsymbol{x}}) \frac{\mu_{n-1,n}(z_{n-1}, x_n)}{\beta} \frac{\alpha - \hat{I}(\hat{\boldsymbol{\mu}}, \hat{\boldsymbol{z}})}{\alpha}$ |
| | | 1 | $\hat{p}(\hat{\boldsymbol{x}}) \frac{\mu_{n-1,n}(z_{n-1}, z_n)}{\beta}$ |
| $\forall\, l\ \ x_l = z_l$ | 1 | 0 | $\hat{I}(\hat{\boldsymbol{\mu}}, \hat{\boldsymbol{z}}) \frac{\mu_{n-1,n}(z_{n-1}, x_n)}{\alpha}$ |
| | | 1 | 0 |

which indeed marginalizes to $\boldsymbol{\mu}$, and $p(\boldsymbol{z}) = 0$ so that $p_{\boldsymbol{\mu}}(\boldsymbol{z}) = 0$, as required.  $\square$

## Footnotes

[1]This is equivalent to finding the maximum probability assignments for a model $p(\boldsymbol{x}) \propto e^{f(\boldsymbol{x}; \theta)}$.

[2]We use the notation $\boldsymbol{\mu} \cdot \boldsymbol{\theta} = \sum_{ij \in E} \sum_{x_i, x_j} \mu_{ij}(x_i, x_j) \theta_{ij}(x_i, x_j) + \sum_i \sum_{x_i} \mu_i(x_i) \theta_i(x_i)$

[3]Consider the case of a single edge between 2 nodes where the MAP assignment is $(0, 0)$. Adding the inequality $\mu_1(0) + \mu_2(0) \leq 1$ produces the fractional vertex $(0.5, 0.5)$.

[4]The connection to Bethe can be more clearly understood from a duality-based proof of Theorem 1. We will cover this in an extended version of the manuscript.

[5]Note that $\hat{\mathcal{M}}(G, \boldsymbol{z}) \subseteq \hat{\mathcal{M}}_L^{\mathcal{ST}}(G, \boldsymbol{z}) \subset \mathcal{M}_L(G)$.

[6]This is very different from the second best constraint, since setting $x_1 = 1$ blocks *all assignments* with this value, as opposed to setting $\boldsymbol{x} = 1$ which blocks only the assignment with all ones.

[7]For BMMF, we used the C implementation at `http://www.cs.huji.ac.il/~talyam/inference.html`. The LPs for STRIPES and Nilsson were solved using CPLEX.

# References

[1] F. Barahona. On cuts and matchings in planar graphs. *Math. Program.*, 60(1):53–68, 1993.

[2] M. Fromer and C. Yanover. Accurate prediction for atomic-level protein design and its application in diversifying the near-optimal sequence space. *Proteins: Structure, Function, and Bioinformatics*, 75:682–705, 2009.

[3] A. Globerson and T. Jaakkola. Fixing max-product: Convergent message passing algorithms for MAP LP-relaxations. In J. Platt, D. Koller, Y. Singer, and S. Roweis, editors, *Advances in Neural Information Processing Systems 21*. MIT Press, Cambridge, MA, 2007.

[4] E. Kloppmann, G. M. Ullmann, and T. Becker. An extended dead-end elimination algorithm to determine gap-free lists of low energy states. *Journal of Comp. Chem.*, 28:2325–2335, 2007.

[5] N. Komodakis and N. Paragios. Beyond loose LP-relaxations: Optimizing MRFs by repairing cycles. In D. Forsyth, P. Torr, and A. Zisserman, editors, *ECCV*, pages 806–820, Heidelberg, Germany, 2008. Springer.

[6] E. L. Lawler. A procedure for computing the K best solutions to discrete optimization problems and its application to the shortest path problem. *Management Science*, 18(7):401–405, 1972.

[7] I. Ljubic, R. Weiskircher, U. Pferschy, G. W. Klau, P. Mutzel, and M. Fischetti. An algorithmic framework for the exact solution of the prize-collecting steiner tree problem. *Mathematical Programming*, 105:427–449, Feb 2006.

[8] D. Nilsson. An efficient algorithm for finding the M most probable configurations in probabilistic expert systems. *Statistics and Computing*, 8:159–173, Jun 1998.

[9] P. Ravikumar, A. Agarwal, and M. Wainwright. Message-passing for graph-structured linear programs: proximal projections, convergence and rounding schemes. In *Proc. of the 25th international conference on Machine learning*, pages 800–807, New York, NY, USA, 2008. ACM.

[10] E. Santos. On the generation of alternative explanations with implications for belief revision. In *Proc. of the 7th Annual Conference on Uncertainty in Artificial Intelligence*, 1991.

[11] Y. Shimony. Finding the MAPs for belief networks is NP-hard. *Aritifical Intelligence*, 68(2):399–410, 1994.

[12] D. Sontag and T. Jaakkola. New outer bounds on the marginal polytope. In J. Platt, D. Koller, Y. Singer, and S. Roweis, editors, *Advances in Neural Information Processing Systems 20*, pages 1393–1400. MIT Press, Cambridge, MA, 2007.

[13] D. Sontag, T. Meltzer, A. Globerson, T. Jaakkola, and Y. Weiss. Tightening LP relaxations for MAP using message passing. In *Proc. of the 24th Annual Conference on Uncertainty in Artificial Intelligence*, pages 503–510, 2008.

[14] B. Taskar, S. Lacoste-Julien, and M. I. Jordan. Structured prediction, dual extragradient and bregman projections. *J. Mach. Learn. Res.*, 7:1627–1653, 2006.

[15] M. Wainwright and M. Jordan. Graphical models, exponential families, and variational inference. *Found. Trends Mach. Learn.*, 1(1-2):1–305, 2008.

[16] M. J. Wainwright, T. Jaakkola, and A. S. Willsky. A new class of upper bounds on the log partition function. *IEEE Transactions on Information Theory*, 51(7):2313–2335, 2005.

[17] T. Werner. A linear programming approach to max-sum problem: A review. *IEEE Trans. Pattern Anal. Mach. Intell.*, 29(7):1165–1179, 2007.

[18] C. Yanover, T. Meltzer, and Y. Weiss. Linear programming relaxations and belief propagation – an empirical study. *Journal of Machine Learning Research*, 7:1887–1907, 2006.

[19] C. Yanover and Y. Weiss. Finding the M most probable configurations using loopy belief propagation. In *Advances in Neural Information Processing Systems 16*. MIT Press, Cambridge, MA, 2004.

[20] J. Yedidia, W. W.T. Freeman, and Y. Weiss. Constructing free-energy approximations and generalized belief propagation algorithms. *IEEE Trans. on Information Theory*, 51(7):2282–2312, 2005.

